# Efficient Pattern Recognition Using a New Transformation Distance

Patrice Simard      Yann Le Cun      John Denker

AT&T Bell Laboratories, 101 Crawford Corner Road, Holmdel, NJ 07724

## Abstract

Memory-based classification algorithms such as radial basis functions or K-nearest neighbors typically rely on simple distances (Euclidean, dot product...), which are not particularly meaningful on pattern vectors. More complex, better suited distance measures are often expensive and rather ad-hoc (elastic matching, deformable templates). We propose a new distance measure which (a) can be made locally invariant to any set of transformations of the input and (b) can be computed efficiently. We tested the method on large handwritten character databases provided by the Post Office and the NIST. Using invariances with respect to translation, rotation, scaling, shearing and line thickness, the method consistently outperformed all other systems tested on the same databases.

## 1   INTRODUCTION

Distance-based classification algorithms such as radial basis functions or K-nearest neighbors often rely on simple distances (such as Euclidean distance, Hamming distance, etc.). As a result, they suffer from a very high sensitivity to simple transformations of the input patterns that should leave the classification unchanged (e.g. translation or scaling for 2D images). This is illustrated in Fig. 1 where an unlabeled image of a "9" must be classified by finding the closest prototype image out of two images representing respectively a "9" and a "4". According to the Euclidean distance (sum of the squares of the pixel to pixel differences), the "4" is closer even though the "9" is much more similar once it has been rotated and thickened. The result is an incorrect classification. The key idea is to construct a distance measure which is invariant with respect to some chosen transformations such as translation, rotation and others. The special case of linear transformations has been well studied in statistics and is sometimes referred to as Procrustes analysis

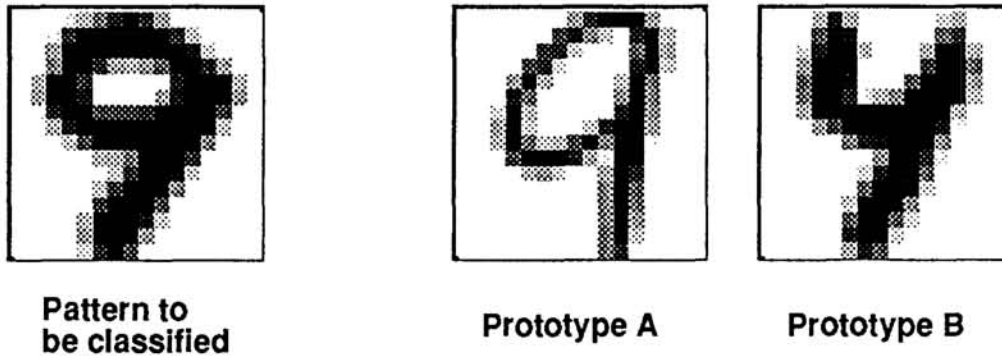

**Pattern to be classified**          **Prototype A**          **Prototype B**

Figure 1: What is a good similarity measure? According to the Euclidean distance the pattern to be classified is more similar to prototype B. A better distance measure would find that prototype A is closer because it differs mainly by a rotation and a thickness transformation, two transformations which should leave the classification invariant.

(Sibson, 1978). It has been applied to on-line character recognition (Sinden and Wilfong, 1992).

This paper considers the more general case of non-linear transformations such as geometric transformations of gray-level images. Remember that even a simple image translation corresponds to a highly non-linear transformation in the high-dimensional pixel space[1]. In previous work (Simard et al., 1992b), we showed how a neural network could be trained to be invariant with respect to selected transformations of the input. We now apply similar ideas to distance-based classifiers.

When a pattern $P$ is transformed (e.g. rotated) with a transformation $s$ that depends on one parameter $\alpha$ (e.g. the angle of the rotation), the set of all the transformed patterns $S_P = \{x \mid \exists \vec{\alpha} \text{ such that } x = s(\vec{\alpha}, P)\}$ is a one-dimensional curve in the vector space of the inputs (see Fig. 2). In certain cases, such as rotations of digitized images, this curve must be made continuous using smoothing techniques (see (Simard et al., 1992b)). When the set of transformations is parameterized by $n$ parameters $\alpha_i$ (rotation, translation, scaling, etc.), $S_P$ is a manifold of at most $n$ dimensions. The patterns in $S_P$ that are obtained through *small* transformations of $P$, i.e. the part of $S_P$ that is close to $P$, can be approximated by a plane tangent to the manifold $S_P$ at the point $P$. Small transformations of $P$ can be obtained by adding to $P$ a linear combination of vectors that span the tangent plane (tangent vectors). The images at the bottom of Fig. 2 were obtained by that procedure. Tangent vectors for a transformation $s$ can easily be computed by finite difference (evaluating $\partial s(\alpha, P)/\partial \alpha$); more details can be found in (Simard et al., 1992b; Simard et al., 1992a).

As we mentioned earlier, the Euclidean distance between two patterns $P$ and $E$ is in general not appropriate because it is sensitive to irrelevant transformations of $P$ and of $E$. In contrast, the distance $\mathcal{D}(E, P)$ defined to be the minimal distance between the two manifolds $S_P$ and $S_E$ is truly invariant with respect to the transformation used to generate $S_P$ and $S_E$. Unfortunately, these manifolds have no analytic expression in general, and finding the distance between them is a hard optimization problem with multiple local minima. Besides, true invariance is not

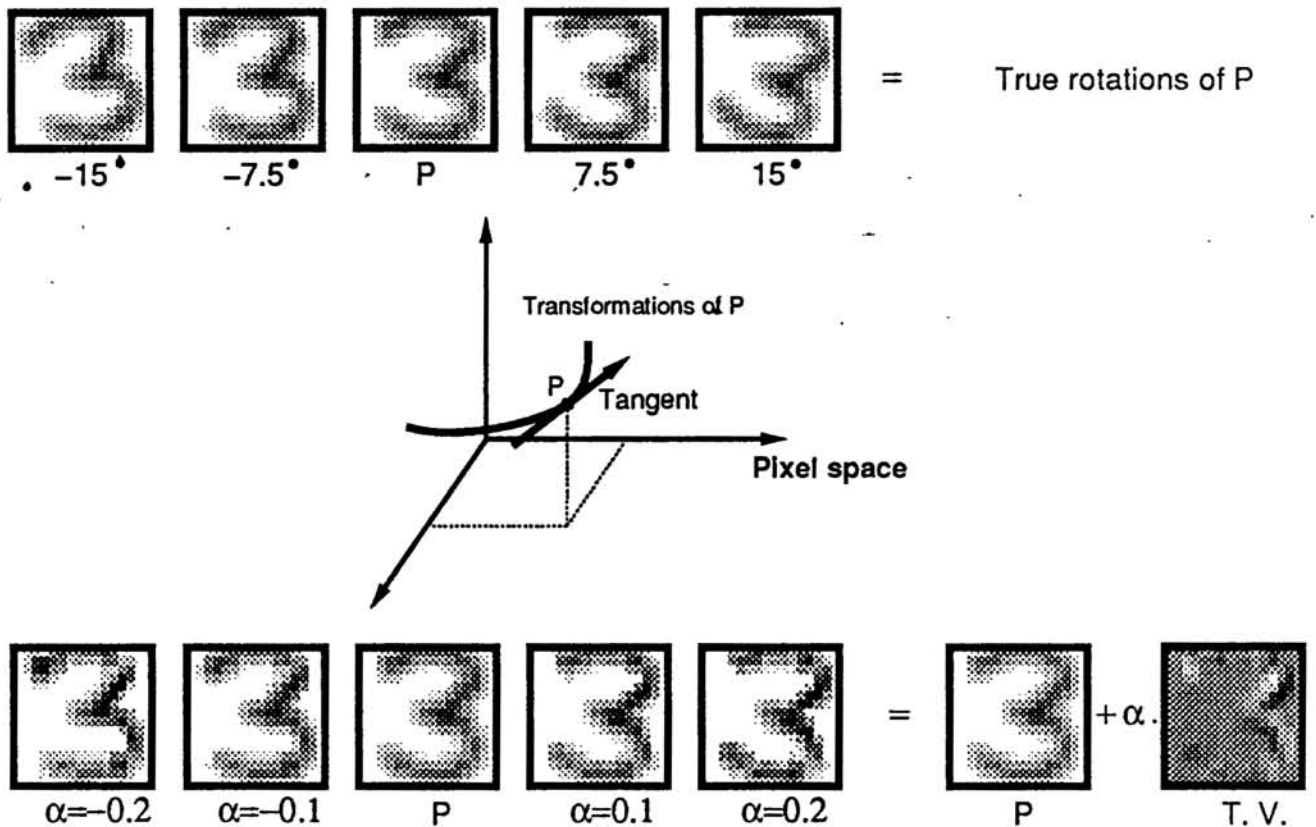

Figure 2: Top: Small rotations of an original digitized image of the digit "3". Middle: Representation of the effect of the rotation in pixel space (if there were only 3 pixels). Bottom: Images obtained by moving along the tangent to the transformation curve for the same original digitized image $P$ by adding various amounts ($\alpha$) of the tangent vector (T.V.).

necessarily desirable since a rotation of a "6" into a "9" does not preserve the correct classification.

Our approach consists of approximating the non-linear manifold $S_P$ and $S_E$ by linear surfaces and computing the distance $D(E, P)$ defined to be the minimum distance between them. This solves three problems at once: 1) linear manifolds have simple analytical expressions which can be easily computed and stored, 2) finding the minimum distance between linear manifolds is a simple least squares problem which can be solved efficiently and, 3) this distance is locally invariant but not globally invariant. Thus the distance between a "6" and a slightly rotated "6" is small but the distance between a "6" and a "9" is large. The different distances between $P$ and $E$ are represented schematically in Fig. 3.

The figure represents two patterns $P$ and $E$ in 3-dimensional space. The manifolds generated by $s$ are represented by one-dimensional curves going through $E$ and $P$ respectively. The linear approximations to the manifolds are represented by lines tangent to the curves at $E$ and $P$. These lines do not intersect in 3 dimensions and the shortest distance between them (uniquely defined) is $D(E, P)$. The distance between the two non-linear transformation curves $\mathcal{D}(E, P)$ is also shown on the figure.

An efficient implementation of the tangent distance $D(E, P)$ will be given in the

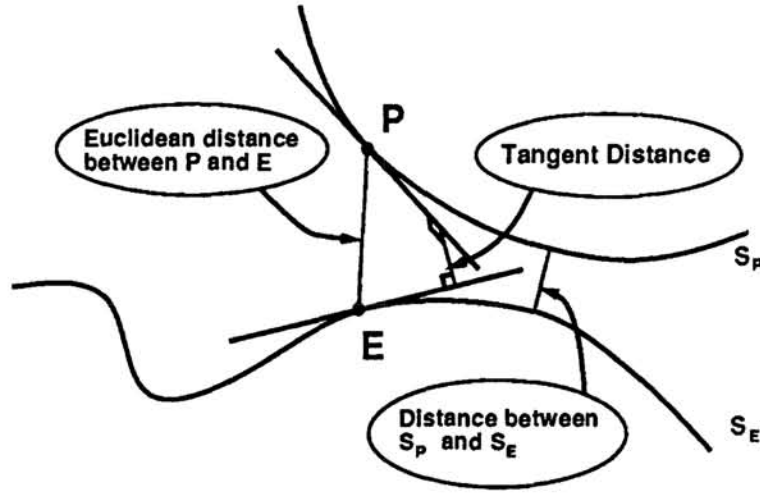

Figure 3: Illustration of the Euclidean distance and the tangent distance between $P$ and $E$

next section. Although the tangent distance can be applied to any kind of patterns represented as vectors, we have concentrated our efforts on applications to image recognition. Comparison of tangent distance with the best known competing method will be described. Finally we will discuss possible variations on the tangent distance and how it can be generalized to problems other than pattern recognition.

## 2  IMPLEMENTATION

In this section we describe formally the computation of the tangent distance. Let the function $s$ which map $u, \vec{\alpha}$ to $s(\vec{\alpha}, u)$ be a differentiable transformation of the input space, depending on a vector $\vec{\alpha}$ of parameter, verifying $s(\vec{0}, u) = u$.

If $u$ is a 2 dimensional image for instance, $s(\vec{\alpha}, u)$ could be a rotation of $u$ by the angle $\vec{\alpha}$. If we are interested in all transformations of images which conserve distances (isometry), $s(\vec{\alpha}, u)$ would be a rotation by $\alpha_r$ followed by a translation by $\alpha_x, \alpha_y$ of the image $u$. In this case $\vec{\alpha} = (\alpha_r, \alpha_x, \alpha_y)$ is a vector of parameters of dimension 3. In general, $\vec{\alpha} = (\alpha_0, \ldots, \alpha_{m-1})$ is of dimension $m$.

Since $s$ is differentiable, the set $S_u = \{x \mid \exists \vec{\alpha} \text{ for which } x = s(\vec{\alpha}, u)\}$ is a differentiable manifold which can be approximated to the first order by a hyperplane $T_u$. This hyperplane is tangent to $S_u$ at $u$ and is generated by the columns of matrix

$$L_u = \left. \frac{\partial s(\vec{\alpha}, u)}{\partial \vec{\alpha}} \right|_{\vec{\alpha}=\vec{0}} = \left[ \frac{\partial s(\vec{\alpha}, u)}{\partial \alpha_0}, \ldots, \frac{\partial s(\vec{\alpha}, u)}{\partial \alpha_{m-1}} \right]_{\vec{\alpha}=\vec{0}} \quad (1)$$

which are vectors tangent to the manifold. If $E$ and $P$ are two patterns to be compared, the respective tangent planes $T_E$ and $T_P$ can be used to define a new distance $D$ between these two patterns. The tangent distance $D(E, P)$ between $E$ and $P$ is defined by

$$D(E, P) = \min_{x \in T_E, y \in T_P} \|x - y\|^2 \quad (2)$$

The equation of the tangent planes $T_E$ and $T_P$ is given by:

$$E'(\vec{\alpha}_E) = E + L_E \vec{\alpha}_E \quad (3)$$
$$P'(\vec{\alpha}_P) = P + L_P \vec{\alpha}_P \quad (4)$$

where $L_E$ and $L_P$ are the matrices containing the tangent vectors (see Eq. 1) and the vectors $\vec{\alpha}_E$ and $\vec{\alpha}_P$ are the coordinates of $E'$ and $P'$ in the corresponding tangent planes. The quantities $L_E$ and $L_P$ are attributes of the patterns so in many cases they can be precomputed and stored.

Computing the tangent distance

$$D(E, P) = \min_{\vec{\alpha}_E, \vec{\alpha}_P} \|E'(\vec{\alpha}_E) - P'(\vec{\alpha}_P)\|^2 \tag{5}$$

amounts to solving a linear least squares problem. The optimality condition is that the partial derivatives of $D(E, P)$ with respect to $\vec{\alpha}_P$ and $\vec{\alpha}_E$ should be zero:

$$\frac{\partial D(E, P)}{\partial \vec{\alpha}_E} = 2(E'(\vec{\alpha}_E) - P'(\vec{\alpha}_P))^\mathsf{T} L_E = 0 \tag{6}$$

$$\frac{\partial D(E, P)}{\partial \vec{\alpha}_P} = 2(P'(\vec{\alpha}_P) - E'(\vec{\alpha}_E))^\mathsf{T} L_P = 0 \tag{7}$$

Substituting $E'$ and $P'$ by their expressions yields to the following linear system of equations, which we must solve for $\vec{\alpha}_P$ and $\vec{\alpha}_E$:

$$L_P^\mathsf{T}(E - P - L_P\vec{\alpha}_P + L_E\vec{\alpha}_E) = 0 \tag{8}$$

$$L_E^\mathsf{T}(E - P - L_P\vec{\alpha}_P + L_E\vec{\alpha}_E) = 0 \tag{9}$$

The solution of this system is

$$(L_{PE}L_{EE}^{-1}L_E^\mathsf{T} - L_P^\mathsf{T})(E - P) = (L_{PE}L_{EE}^{-1}L_{EP} - L_{PP})\vec{\alpha}_P \tag{10}$$

$$(L_{EP}L_{PP}^{-1}L_P^\mathsf{T} - L_E^\mathsf{T})(E - P) = (L_{EE} - L_{EP}L_{PP}^{-1}L_{PE})\vec{\alpha}_E \tag{11}$$

where $L_{EE} = L_E^\mathsf{T} L_E$, $L_{PE} = L_P^\mathsf{T} L_E$, $L_{EP} = L_E^\mathsf{T} L_P$ and $L_{PP} = L_P^\mathsf{T} L_P$. LU decompositions of $L_{EE}$ and $L_{PP}$ can be precomputed. The most expensive part in solving this system is evaluating $L_{EP}$ ($L_{PE}$ can be obtained by transposing $L_{EP}$). It requires $m_E \times m_P$ dot products, where $m_E$ is the number of tangent vectors for $E$ and $m_P$ is the number of tangent vectors for $P$. Once $L_{EP}$ has been computed, $\vec{\alpha}_P$ and $\vec{\alpha}_E$ can be computed by solving two (small) linear system of respectively $m_E$ and $m_P$ equations. The tangent distance is obtained by computing $\|E'(\vec{\alpha}_E) - P'(\vec{\alpha}_P)\|$ using the value of $\vec{\alpha}_P$ and $\vec{\alpha}_E$ in equations 3 and 4. If $n$ is the length of vector $E$ (or $P$), the algorithm described above requires roughly $n(m_E+1)(m_P+1)+3(m_E^3+m_P^3)$ multiply-adds. Approximations to the tangent distance can be computed more efficiently.

## 3   RESULTS

Before giving the results of handwritten digit recognition experiments, we would like to demonstrate the property of "local invariance" of tangent distance. A 16 by 16 pixel image similar to the "3" in Fig 2 was translated by various amounts. The tangent distance (using the tangent vector corresponding to horizontal translations) and the Euclidean Distance between the original image and its translated version were measured as a function of the size $k$ (in pixels) of the translation. The result is plotted in Fig. 4. It is clear that the Euclidean Distance starts increasing linearly with $k$ while the tangent distance remains very small for translations as large as two pixels. This indicates that, while Euclidean Distance is not invariant to translation, tangent distance is locally invariant. The extent of the invariance can be

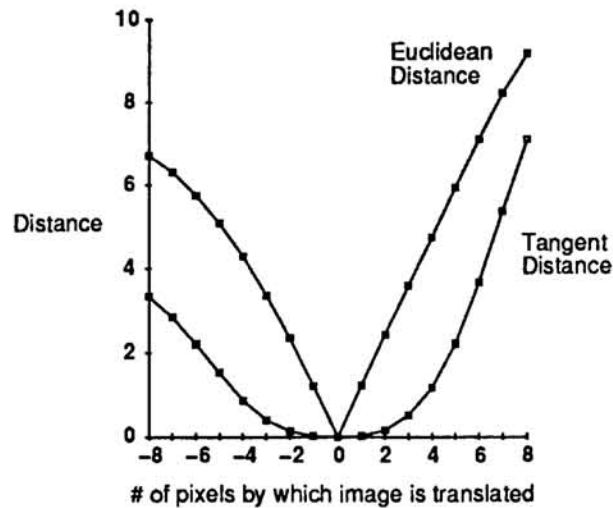

# of pixels by which image is translated

Figure 4: Euclidean and tangent distances between a 16x16 handwritten digit image and its translated version as a function of the amount of translation measured in pixels.

increased by smoothing the original image, but significant features may be blurred away, leading to confusion errors. The figure is not symmetric for large translations because the translated image is truncated to the 16 by 16 pixel field of the original image. In the following experiments, smoothing was done by convolution with a Gaussian of standard deviation $\sigma = 0.75$. This value, which was estimated visually, turned out to be nearly optimal (but not critical).

## 3.1   Handwritten Digit Recognition

Experiments were conducted to evaluate the performance of tangent distance for handwritten digit recognition. An interesting characteristic of digit images is that we can readily identify a set of local transformations which do not affect the identity of the character, while covering a large portion of the set of possible *instances* of the character. Seven such image transformations were identified: X and Y translations, rotation, scaling, two hyperbolic transformations (which can generate shearing and squeezing), and line thickening or thinning. The first six transformations were chosen to span the set of all possible linear coordinate transforms in the image plane (nevertheless, they correspond to highly non-linear transforms in pixel space). Additional transformations have been tried with less success.

The simplest possible use of tangent distance is in a Nearest Neighbor classifier. A set of prototypes is selected from a training set, and stored in memory. When a test pattern is to be classified, the $K$ nearest prototypes (in terms of tangent distance) are found, and the pattern is given the class that has the majority among the neighbors. In our applications, the size of the prototype set is in the neighborhood of 10,000. In principle, classifying a pattern would require computing 10,000 tangent distances, leading to excessive classification times, despite the efficiency of the tangent distance computation. Fortunately, two patterns that are very far apart in terms of Euclidean Distance are likely to be far apart in terms of tangent distance. Therefore we can use Euclidean distance as a "prefilter", and eliminate prototypes that are unlikely to be among the nearest neighbors. We used the following 4-step classification procedure: 1) the Euclidean distance is computed between the test pattern and all the prototypes, 2) The closest 100 prototypes are selected, 3) the tangent distance between these 100 prototypes and the test pattern is computed

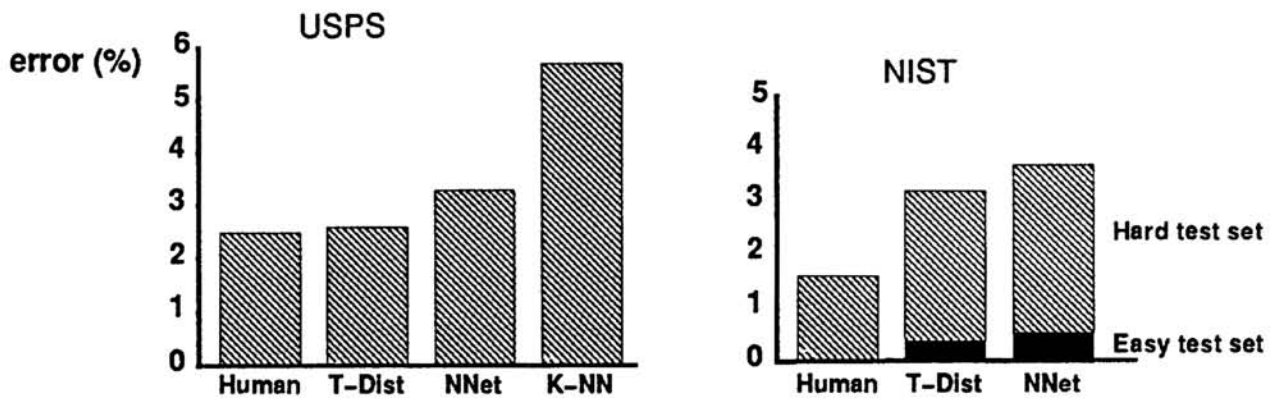

Figure 5: Comparison of the error rate of tangent nearest neighbors and other methods on two handwritten digit databases

and 4) the most represented label among the $K$ closest prototype is outputed. This procedure is two orders of magnitude faster than computing all 10,000 tangent distances, and yields the same performance.

**US Postal Service database:** In the first experiment, the database consisted of 16 by 16 pixel size-normalized images of handwritten digits, coming from US mail envelopes. The entire training set of 9709 examples of was used as the prototype set. The test set contained 2007 patterns. The best performance was obtained with the "one nearest neighbor" rule. The results are plotted in Fig. 5. The error rate of the method is 2.6%. Two members of our group labeled the test set by hand with an error rate of 2.5% (using one of their labelings as the truth to test the other also yielded 2.5% error rate). This is a good indicator of the level of difficulty of this task[2]. The performance of our best neural network (Le Cun et al., 1990) was 3.3%. The performance of one nearest neighbor with the Euclidean distance was 5.9%. These results show that tangent distance performs substantially better than both standard K-nearest neighbor and neural networks.

**NIST database:** The second experiment was a competition organized by the National Institute of Standards and Technology. The object of the competition was to classify a test set of 59,000 handwritten digits, given a training set of 223,000 patterns. A total of 45 algorithms were submitted from 26 companies from 7 different countries. Since the training set was so big, a very simple procedure was used to select about 12,000 patterns as prototypes. The procedure consists of creating a new database (empty at the beginning), and classifying each pattern of the large database using the new database as a prototype set. Each time an error is made, the pattern is added to the new database. More than one pass may have to be made before the new database is stable. Since this filtering process would take too long with 223,000 prototypes, we split the large database into 22 smaller databases of 10,000 patterns each, filtered those (to about 550 patterns) and concatenated the result, yielding a database of roughly 12,000 patterns. This procedure has many drawbacks, and in particular, it is very good at picking up mislabeled characters in the training set. To counteract this unfortunate effect, a 3 nearest neighbors procedure was used with tangent distance. The organizers decided to collect the

training set and the test set among two very different populations (census bureau workers for the training set, high-school students for the test set), we therefore report results on the official NIST test set (named "hard test set"), and on a subset of the official training set, which we kept aside for test purposes (the "easy test set"). The results are shown in Fig. 5. The performance is much worse on the hard test set since the distribution was very different from that of the training set. Out of the 25 participants *who used the NIST training database*, tangent distance finished first. The overall winner did not use the training set provided by NIST (he used a much larger proprietary training set), and therefore was not affected by the different distributions in the training set and test set.

# 4   DISCUSSION

The tangent distance algorithm described in the implementation section can be improved/adjusted in at least four different ways: 1) approximating the tangent distance for better speed 2) modifying the tangent distance itself, 3) changing the set of transformations/tangent vectors and 4) using the tangent distance with classification algorithms other than K-nearest neighbors, perhaps in combination, to minimize the number of prototypes. We will discuss each of these aspects in turn.

**Approximation:** The distance between two hyperplanes $T_E$ and $T_P$ going through $P$ and $E$ can be approximated by computing the projection $\mathcal{P}_E(P)$ of $P$ onto $T_E$ and $\mathcal{P}_P(E)$ of $E$ onto $T_P$. The distance $\|\mathcal{P}_E(P) - \mathcal{P}_P(E)\|$ can be computed in $O(n(m_E + m_P))$ multiply-adds and is a fairly good approximation of $D(E, P)$. This approximation can be improved at very low cost by computing the closest points between the lines defined by $(E, \mathcal{P}_E(P))$ and $(P, \mathcal{P}_P(E))$. This approximation was used with no loss of performance to reduce the number of computed tangent distance from 100 to 20 (this involves an additional "prefilter"). In the case of images, another time-saving idea is to compute tangent distance on progressively smaller sets of progressively higher resolution images.

**Changing the distance:** One may worry that the tangent planes of $E$ and $P$ may be parallel and be very close at a very distant region (a bad side effect of the linear approximation). This effect can be limited by imposing a constraint of the form $\|\vec{\alpha_E}\| < K_E$ and $\|\vec{\alpha_P}\| < K_P$. This constraint was implemented but did not yield better results. The reason is that tangent planes are mostly orthogonal in high dimensional space and the norms of $\|\vec{\alpha_E}\|$ and $\|\vec{\alpha_P}\|$ are already small.

The tangent distance can be normalized by dividing it by the norm of the vectors. This improves the results slightly because it offsets side effects introduced in some transformations such as scaling. Indeed, if scaling is a transformation of interest, there is a potential danger of finding the minimum distance between two images after they have been scaled down to a single point. The linear approximation of the scaling transformation does not reach this extreme, but still yields a slight degradation of the performance. The error rate reported on the USPS database can be improved to 2.4% using this normalization (which was not tried on NIST).

Tangent distance can be viewed as one iteration of a Newton-type algorithm which finds the points of minimum distance on the true transformation manifolds. The vectors $\vec{\alpha_E}$ and $\vec{\alpha_P}$ are the coordinates of the two closest points in the respective tangent spaces, but they can also be interpreted for real (non-linear) transformations. If $\vec{\alpha_{E,i}}$ is the amount of the translation tangent vector that must be added to $E$ to make it as close as possible to $P$, we can compute the true translation of image $E$ by $\vec{\alpha_{E,i}}$ pixels. In other words, $E'(\alpha_E)$ and $P'(\alpha_P)$ are projected onto

close points of $S_E$ and $S_P$. This involves a resampling but can be done efficiently. Once this new image has been computed, the corresponding tangent vectors can be computed for this new image and the process can be repeated. Eventually this will converge to a local minimum in the distance between the two transformation manifold of $P$ and $E$. The tangent distance needs to be normalized for this iteration process to work.

**A priori knowledge:** The *a priori* knowledge used for tangent vectors depends greatly on the application. For character recognition, thickness was one of the most important transformations, reducing the error rate from 3.3% to 2.6%. Such a transformation would be meaningless in, say, speech or face recognition. Other transformations such as local rubber sheet deformations may be interesting for character recognition. Transformations can be known *a priori* or learned from the data.

**Other algorithms, reducing the number of prototypes:** Tangent distance is a general method that can be applied to problems other than image recognition, with classification methods other than K-nearest neighbors. Many distance-based classification schemes could be used in conjunction with tangent distance, among them LVQ (Kohonen, 1984), and radial basis functions. Since all the operators involved in the tangent distance are differentiable, it is possible to compute the partial derivative of the tangent distance (between an object and a prototype) with respect to the tangent vectors, or with respect to the prototype. Therefore the tangent distance operators can be inserted in gradient-descent based adaptive machines (of which LVQ and RBF are particular cases). The main advantage of learning the prototypes or the tangent vectors is that fewer prototypes may be needed to reach the same (or superior) level of performance as, say, regular K-nearest neighbors.

In conclusion, tangent distance can greatly improve many of the distance-based algorithms. We have used tangent distance in the simple K-nearest neighbor algorithm and outperformed all existing techniques on standard classification tasks. This surprising success is probably due the fact that *a priori* knowledge can be very effectively expressed in the form of tangent vectors. Fortunately, many algorithms are based on computing distances and can be adapted to express *a priori* knowledge in a similar fashion. Promising candidates include Parzen windows, learning vector quantization and radial basis functions.

## Footnotes

[1] If the image of a "3" is translated vertically upward, the middle top pixel will oscillate from black to white three times.

[2]This is an extremely difficult test set. Procedures that achieve less than 0.5% error on other handwritten digit tasks barely achieve less than 4% on this one

# References

Kohonen, T. (1984). Self-organization and Associative Memory. In *Springer Series in Information Sciences*, volume 8. Springer-Verlag.

Le Cun, Y., Boser, B., Denker, J. S., Henderson, D., Howard, R. E., Hubbard, W., and Jackel, L. D. (1990). Handwritten digit recognition with a back-propagation network. In Touretzky, D., editor, *Advances in Neural Information Processing Systems 2 (NIPS*89)*, Denver, CO. Morgan Kaufman.

Sibson, R. (1978). Studies in the Robustness of Multidimensional Scaling: Procrustes Statistices. *J. R. Statist. Soc.*, 40:234–238.

Simard, P. Y., LeCun, Y., Denker, J., and Victorri, B. (1992a). An Efficient Method for Learning Invariances in Adaptive classifiers. In *International Conference on Pattern Recognition*, volume 2, pages 651–655, The Hague, Netherlands.

Simard, P. Y., Victorri, B., LeCun, Y., and Denker, J. (1992b). Tangent Prop – A formalism for specifying selected invariances in an adaptive network. In *Neural Information Processing Systems*, volume 4, pages 895–903, San Mateo, CA.

Sinden, F. and Wilfong, G. (1992). On-line Recognition of Handwritten Symbols. Technical Report 11228-910930-02IM, AT&T Bell Laboratories.
